# Digital-Analog Hybrid Synapse Chips for Electronic Neural Networks

A. Moopenn, T. Duong, and A.P. Thakoor
Center for Space Microelectronics Technology
Jet Propulsion Laboratory/California Institute of Technology
Pasadena, CA 91109

## ABSTRACT

Cascadable, CMOS synapse chips containing a cross-bar array of 32x32 (1024) programmable synapses have been fabricated as "building blocks" for fully parallel implementation of neural networks. The synapses are based on a hybrid digital-analog design which utilizes on-chip 7-bit data latches to store quantized weights and two-quadrant multiplying DAC's to compute weighted outputs. The synapses exhibit 6-bit resolution and excellent monotonicity and consistency in their transfer characteristics. A 64-neuron hardware incorporating four synapse chips has been fabricated to investigate the performance of feedback networks in optimization problem solving. In this study, a 7x7, one-to-one assignment net and the Hopfield-Tank 8-city traveling salesman problem net have been implemented in the hardware. The network's ability to obtain optimum or near optimum solutions in real time has been demonstrated.

## 1 INTRODUCTION

A large number of electrically modifiable synapses is often required for fully parallel analog neural network hardware. Electronic synapses based on CMOS, EEPROM, as well as thin film technologies are actively being developed [1-5]. One preferred approach is based on a hybrid digital-analog design which can easily be implemented in CMOS with simple interface and analog circuitry. The hybrid design utilizes digital memories to store the synaptic weights and digital-to-analog converters to perform analog multiplication. A variety of synaptic chips based on such hybrid designs have been developed and used as "building blocks" in larger neural network hardware systems fabricated at JPL.

In this paper, the design and operational characteristics of the hybrid synapse chips are described. The development of a 64-neuron hardware incorporating several of

the synapse chips is also discussed. Finally, a hardware implementation of two global optimization nets, namely, the one-to-one assignment optimization net and the Hopfield-Tank traveling salesman net [6], and their performance based on our 64-neuron hardware are discussed.

## 2 CHIP DESIGN AND ELECTRICAL CHARACTERISTICS

The basic design and operational characteristics of the hybrid digital analog synapse chips are described in this section. A simplified block diagram of the chips is shown in Fig. 1. The chips consist of an address/data de-multiplexer, row and column address decoders, 64 analog input/output lines, and 1024 synapse cells arranged in the form of a 32x32 cross-bar matrix. The synapse cells along the i-th row have a common output, $x_i$, and similarly, synapses along the j-th column have a common input, $y_j$. The synapse input/output lines are brought off-chip for multi-chip expansion to a larger synaptic matrix. The synapse cell, based on a hybrid digital analog design, essentially consists of a 7-bit static latch and a 7-bit, two-quadrant multiplying DAC.

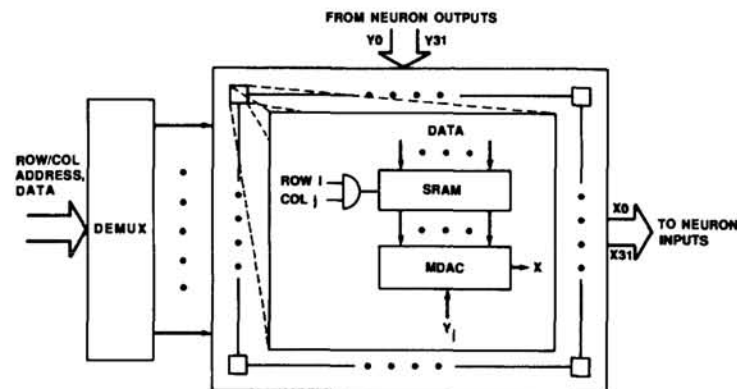

**Figure 1:** Simplified block diagram of hybrid 32x32x7-bit synapse chip.

A circuit diagram of the 7-bit DAC is shown in Fig. 2. The DAC consists of a current input circuit, a set of binary weighted current sources, and a current steering circuit. The current in the input circuit is mirrored by the binary-weighted current sources for all synapses along a column. In one version of the chips, a single long-channel FET is used to convert the synapse input voltage to a current. In addition, the gate of the transistor is connected internally to the gates of other long channel transistors. This common gate is accessible off-chip and provides a means for controlling the overall "gain" of the synapses in the chip. In a second chip version, an external resistor is employed to perform input voltage to current conversion when a high linearity in the synapse transfer characteristics is desired.

Hybrid 32x32x7-bit synapse chips with and without long channel transistors were fabricated through MOSIS using a 2-micron, n-well CMOS process. Typical measured synapse response (I-V) curves from these chips are shown in Figs. 3a and 3b for weight values of 0, +/- 1, 3, 7, 15, 31, and 63. The curves in Fig. 3a were obtained for a synapse incorporating an on-chip long-channel FET with a gate bias of 5 volts. The non-linear synapse response is evident and can be seen to be similar to that of a "threshold" current source. The non-linear behavior is mainly attributed to the nonlinear drain characteristics of the long channel transistor. It should be pointed out that synapses with such characteristics are especially suited for neural networks with neurons operating in the high gain limit, in which case, the nonlinearity may even be desirable. The set of curves in Fig. 3b were obtained using an external 10-megaohm resistor for the V-I conversion. For input voltages greater than about twice the transistor's threshold voltage ($\sim 0.8$ v), the synapse's current output is a highly linear function of the input voltage. The linear characteristics achieved with the use of external resistors would be applicable in feedforward nets with learning capabilities.

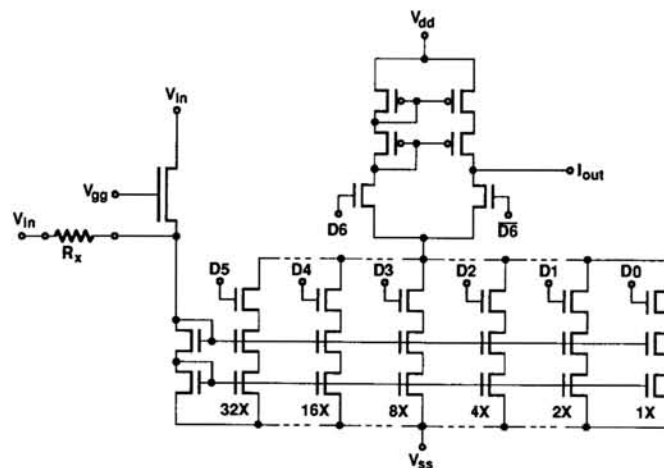

Figure 2:  Circuit diagram of 7-bit multiplying DAC.

Figure 4 shows the measured output of the synapse as the weight is incremented from -60 to +60. The synapse exhibits excellent monotonicity and step size consistency. Based on a random sampling of synapses from several chips, the step size standard deviation due to mismatched transistor characteristics is typically less than 25 percent.

## 3   64-NEURON HARDWARE

The hybrid synapse chips are ideally suited for hardware implementation of feedback neural networks for combinatorial global optimization problem solving or associative recall where the synaptic weights are known a priori. For example, in a Hopfield-type feedback net [7], the weights can be calculated directly from a set of cost parameters or a set of stored vectors. The desired weights are

quantized and downloaded into the memories of the synapse chips. On the other hand, in supervised learning applications, learning can be performed off-line, taking into consideration the operating characteristics of the synapses, and the new updated weights are simply reprogrammed into the synaptic hardware during each training cycle.

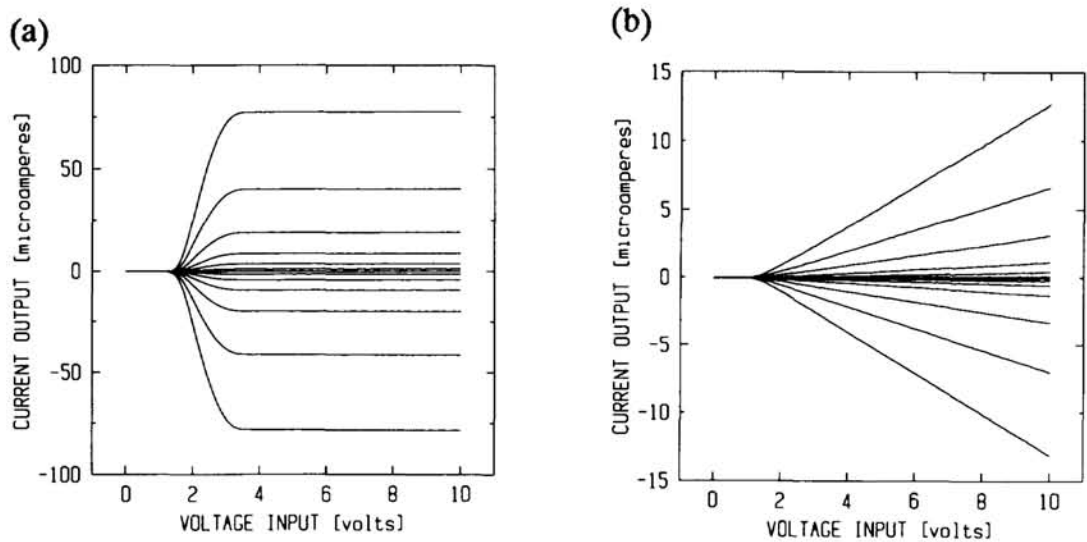

**Figure 3:** Transfer characteristics of a 7-bit synapse for weight values of 0, +/- 1, 3, 7, 15, 31, 63, (a) with long channel transistors for voltage to current conversion ($V_{gg}$= 5.0 volts) and (b) with external 10 mega-ohm resistor.

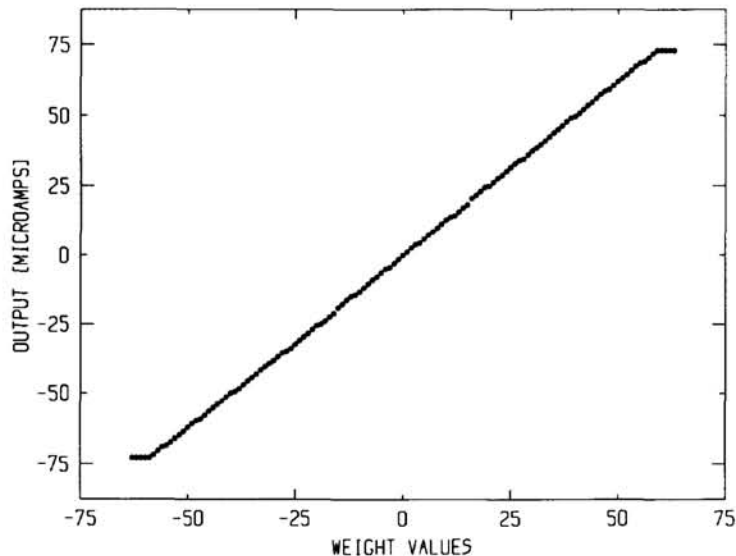

**Figure 4:** Synapse output as weight value is incremented from -60 to +60. ($V_{gg}$=$V_{in}$= 5.0 volts)

A 64-neuron breadboard system incorporating several of the hybrid synapse chips has been fabricated to demonstrate the utility of these building block chips, and to investigate the dynamical properties, global optimization problem solving abilities, and application potential of neural networks. The system consists of an array of 64 discrete neurons and four hybrid synapse chips connected to form a 64x64 cross-bar synapse matrix. Each neuron is an operational-amplifier operating as a current summing amplifier. A circuit model of a neuron with some synapses is shown in Fig. 5. The system dynamical equations are given by:

$$\tau_f \, dV_i/dt = \Sigma \, T_{ij} \, V_j - V_i + R_f \, I_i,$$

where $V_i$ is the output of the neuron i, $T_{ij}$ is the synaptic weight from neuron j to neuron i, $R_f$ and $C_f$ are the feedback resistance and capacitance of the neuron, $\tau_f = R_f \, C_f$, and $I_i$ is the external input current. For our system, $R_f$ was about 50 kilo-ohms, and $C_f$ was about 10 pF, a value large enough to ensure stability against oscillations. The system was interfaced to a microcomputer which allows downloading of the synaptic weight data and analog readout of the neuron states.

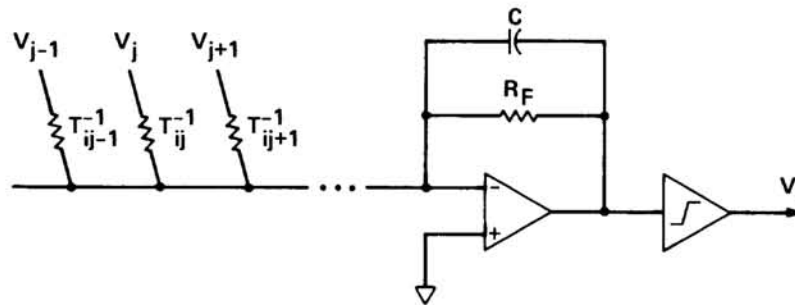

**Figure 5:** Electronic circuit model of neuron and synapses.

## 4 GLOBAL OPTIMIZATION NEURAL NETS

Two combinatorial global optimization problems, namely, the one-to-one assignment problem and the traveling salesman problem, were selected for our neural net hardware implementation study. Of particular interest is the performance of the optimization network in terms of the quality and speed of solutions in light of hardware limitations.

In the one-to-one assignment problem, given two sets of N elements and a cost assignment matrix, the objective is to assign each element in one set to an element in the second set so as to minimize the total assignment cost. In our neural net implementation, the network is a Hopfield-type feedback net consisting of an NxN array of assignment neurons. In this representation, a permissible set of one-to-one assignments corresponds to a permutation matrix. Thus, lateral inhibition

between assignment neurons is employed to ensure that there is only one active neuron in each row and in each column of the neuron array. To force the network to favor assignment sets with low total assignment cost, each assignment neuron is also given an analog prompt, that is, a fixed analog excitation proportional to a positive constant minus its assignment cost.

An 8-city Hopfield-Tank TSP net was implemented in the 64-neuron hardware. Convergence statistics were similarly obtained from 100 randomly generated 8-city positions. The network was observed to give good solutions using a large synapse gain (common gate bias = 7 volts) and an annealing time of about one neuron time constant ($\sim$ 50 usec). As shown in Fig. 6b, the TSP net found tours which were in the best 6%. It gave the best tours in 11% of the cases and the first to third best tours in 31% of the cases. Although these results are quite good, the performance of the TSP net compares less favorably with the assignment net. This can be expected due to the increased complexity of the TSP net. Furthermore, since the initial state is arbitrary, the TSP net is more likely to settle into a local minimum before reaching the global minimum. On the other hand, in the assignment net, the analog prompt helps to establish an initial state which is close to the global minimum, thereby increasing its likelihood of converging to the optimum solution.

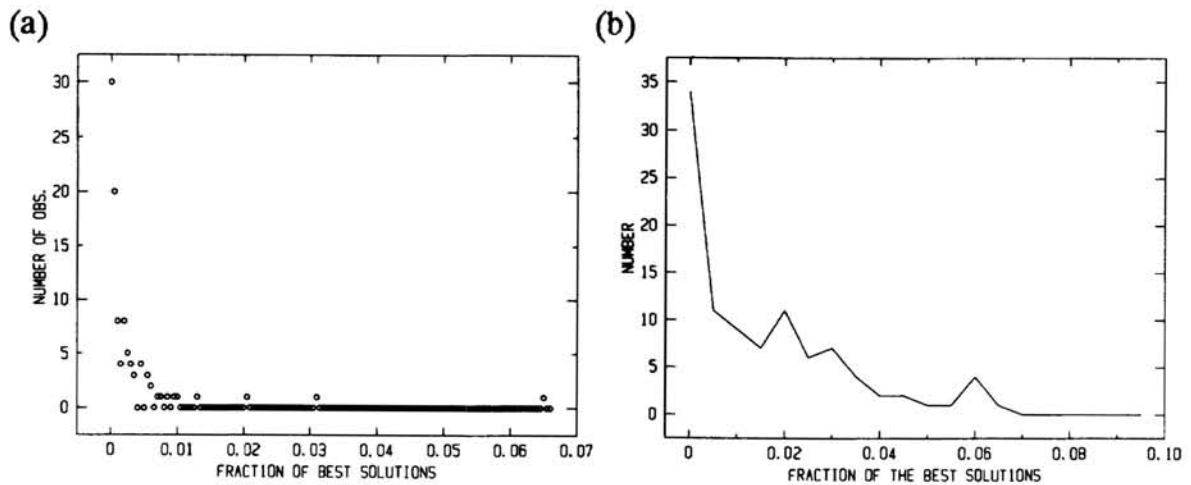

**Figure 6:** Performance statistics for (a) 7x7 assignment problem and (b) 8-city traveling salesman problem.

## 5 CONCLUSIONS

CMOS synapse chips based on a hybrid analog-digital design are ideally suited as building blocks for the development of fully parallel and analog neural net hardware. The chips described in this paper feature 1024 synapses arranged in a 32x32 cross-bar matrix with 120 programmable weight levels for each synapse. Although limited by the process variation in the chip fabrication, a 6-bit weight resolution is achieved with our design. A 64-neuron hardware incorporating several

of the synapse chips is fabricated to investigate the performance of feedback networks in optimization problem solving. The ability of such networks to provide optimum or near optimum solutions to the one-to-one assignment problem and the traveling salesman problem is demonstrated in hardware. The neural hardware is capable of providing real time solutions with settling times in the 50-500 usec In an energy function description, all valid assignment sets correspond to energy minima of equal depth located at corners of the NxN dimensional hypercube (in the large neuron gain limit). The analog prompt term in the energy function has the effect of "tilting" the energy surface toward the hypercube corners with low total assignment cost. Thus, the assignment net may be described as a first-order global optimization net because the analog cost parameters appear only in the linear term of the energy function, i.e., the analog information simply appears as fixed biases and the interaction between neurons is of a binary nature. Since the energy surface contains a large number of local energy minima ($\sim$ N!) there is the strong possibility that the network will get trapped in a local minimum, depending on its initial state. Simulated annealing can be used to reduce this likelihood. One approach is to start with very low neuron gain, and increasing it slowly as the network evolves to a stable state. An alternative but similar approach which can easily be implemented with the current hybrid synapse chips is to gradually increase the synapse gain.

A 7x7 one-to-one assignment problem was implemented in the 64-neuron hardware to investigate the performance of the assignment optimization net. An additional neuron was used to provide the analog biases (quantized to 6 bits) to the assignment neurons. Convergence statistics were obtained from 100 randomly generated cost assignment matrices. For each cost matrix, the synapse gain and annealing time were optimized and the solution obtained by the hardware was recorded. The network generally performed well with a large synapse gain (common gate bias of 7 volts) and an annealing time of about 10 neuron time constants ($\sim$ 500 usec). The unusually large anneal time observed emphasizes the importance of suppressing the quadratic energy term while maintaining the analog prompt in the initial course of the network's state trajectory. Solution distributions for each cost matrix were also obtained from a computer search for the purpose of rating the hardware solutions. The performance of the assignment net is summarized in Fig. 6. In all cases, the network obtained solutions which were in the best 1%. Moreover, the best solutions were obtained in 40% of the cases, and the first, second, third best in 75% of the cases. These results are very encouraging in spite of the limited resolution of the analog biases and the fact that the analog biases also vary in time with the synapse gain.

The Hopfield-Tank's traveling salesman problem (TSP) network [6] was also investigated in the 64-neuron hardware. In this implementation, the analog cost information (i.e., the inter-city distances) is encoded in the connection strength of the synapses. Lateral inhibition is provided via binary synapses to ensure a valid city tour. However, the intercity distance provides additional interaction between

neurons via excitatory synapses with strength proportional to a positive constant minus the distance.  Thus the TSP net, considerably more complex than the assignment net, may be described as a second order global optimization net.
range, which can be further reduced to 1-10 usec with the incorporation of on-chip neurons.

## Acknowledgements

The work described in this paper was performed by the Center for Space Microelectronics Technology, Jet Propulsion Laboratory, California Institute of Technology, and was sponsored in part by the Joint Tactical Fusion Program Office and the Defense Advanced Research Projects Agency, through an agreement with the National Aeronautics and Space Administration.  The authors thank John Lambe and Assad Abidi for many useful discussions, and Tim Shaw for his valuable assistance in the chip-layout design.

## References

1.    S. Eberhardt, T. Duong, and A. Thakoor, "A VLSI Analog Synapse 'Building Block' Chip for Hardware Neural Network Implementations," Proc. IEEE 3rd Annual Parallel Processing Symp., Fullerton, ed. L.H. Canter, vol. 1, pp. 257-267, Mar. 29-31, 1989.

2.    A. Moopenn, A.P. Moopenn, and T. Duong, "Digital-Analog-Hybrid Neural Simulator: A Design Aid for Custom VLSI Neurochips," Proc. SPIE Conf. High Speed Computing, Los Angeles, ed. Keith Bromley, vol. 1058, pp. 147-157, Jan. 17-18, 1989.

3.    M. Holler, S. Tam, H. Castro, R. Benson, "An Electrically Trainable Artificial Neural Network (ETANN) with 10240 'Floating Gate' Synapses," Proc. IJCNN, Wash. D.C., vol. 2, pp. 191-196, June 18-22, 1989.

4.    A.P. Thakoor, A. Moopenn, J. Lambe, and S.K. Khanna, "Electronic Hardware Implementations of Neural Networks," Appl. Optics, vol. 26, no. 23, 1987, pp. 5085-5092.

5.    S. Thakoor, A. Moopenn, T. Daud, and A.P. Thakoor, "Solid State Thin Film Memistor for Electronic Neural Networks," J. Appl. Phys. 1990 (in press).

6.    J.J. Hopfield and D.W. Tank, "Neural Computation of Decisions in Optimization Problems," Biol. Cybern., vol. 52, pp. 141-152, 1985.

7.    J.J. Hopfield, "Neurons with Graded Response Have Collective Computational Properties Like Those of Two-State Neurons," Proc. Nat'l Acad. Sci., vol. 81, 1984, pp. 3088-3092.